# Periodic Step-Size Adaptation for Single-Pass On-line Learning

**Chun-Nan Hsu**[1,2,*]**, Yu-Ming Chang**[1]**, Han-Shen Huang**[1] **and Yuh-Jye Lee**[3]
[1]Institute of Information Science, Academia Sinica, Taipei 115, Taiwan
[2]USC/Information Sciences Institute, Marina del Rey, CA 90292, USA
[3]Department of Computer Science and Information Engineering,
National Taiwan University of Science and Technology, Taipei 106, Taiwan
*chunnan@isi.edu

## Abstract

It has been established that the second-order stochastic gradient descent (2SGD) method can potentially achieve generalization performance as well as empirical optimum in a single pass (i.e., epoch) through the training examples. However, 2SGD requires computing the inverse of the Hessian matrix of the loss function, which is prohibitively expensive. This paper presents *Periodic Step-size Adaptation* (PSA), which approximates the Jacobian matrix of the mapping function and explores a linear relation between the Jacobian and Hessian to approximate the Hessian periodically and achieve near-optimal results in experiments on a wide variety of models and tasks.

## 1   Introduction

On-line learning has been studied for decades. Early works concentrate on minimizing the required number of model corrections made by the algorithm through a single pass of training examples. More recently, on-line learning is considered as a solution of large scale learning mainly because of its fast convergence property. New on-line learning algorithms for large scale learning, such as SMD [1] and EG [2], are designed to learn incrementally to achieve fast convergence. They usually still require several passes (or epochs) through the training examples to converge at a satisfying model. However, the real bottleneck of large scale learning is I/O time. Reading a large data set from disk to memory usually takes much longer than CPU time spent in learning. Therefore, the study of on-line learning should focus more on single-pass performance. That is, after processing all available training examples once, the learned model should generalize as well as possible so that used training example can really be removed from memory to minimize disk I/O time. In natural learning, single-pass learning is also interesting because it allows for continual learning from unlimited training examples under the constraint of limited storage, resembling a nature learner.

Previously, many authors, including [3] and [4], have established that given a sufficiently large set of training examples, 2SGD can potentially achieve generalization performance as well as empirical optimum in a single pass through the training examples. However, 2SGD requires computing the inverse of the Hessian matrix of the loss function, which is prohibitively expensive. Many attempts to approximate the Hessian have been made. For example, one may consider to modify L-BFGS [5] for online settings. L-BFGS relies on line search. But in online settings, we only have the surface of the loss function given one training example, as opposed to all in batch settings. The search direction obtained by line search on such a surface rarely leads to empirical optimum. A review of similar attempts can be found in Bottou's tutorial [6], where he suggested that none is actually sufficient to achieve theoretical single-pass performance in practice. This paper presents a new 2SGD method, called *Periodic Step-size Adaptation* (PSA). PSA approximates the Jacobian matrix of the mapping function and explores a linear relation between the Jacobian and Hessian to approximate the Hessian

periodically. The per-iteration time-complexity of PSA is linear to the number of nonzero dimensions of the data. We analyze the accuracy of the approximation and derive the asymptotic rate of convergence for PSA. Experimental results show that for a wide variety of models and tasks, PSA is always very close to empirical optimum in a single-pass. Experimental results also show that PSA can run much faster compared to state-of-the-art algorithms.

## 2 Aitken's Acceleration

Let $\mathbf{w} \in \mathbb{R}^d$ be a $d$-dimensional weight vector of a model. A machine learning problem can be formulated as a fixed-point iteration that solves the equation $\mathbf{w} = \mathcal{M}(\mathbf{w})$, where $\mathcal{M}$ is a mapping $\mathcal{M} : \mathbb{R}^d \to \mathbb{R}^d$, until $\mathbf{w}^* = \mathcal{M}(\mathbf{w}^*)$. Assume that the mapping $\mathcal{M}$ is differentiable. Then we can apply *Aitken's acceleration*, which attempts to extrapolate to the local optimum in one step, to accelerate the convergence of the mapping:

$$\mathbf{w}^* = \mathbf{w}^{(t)} + (\mathbf{I} - \mathbf{J})^{-1}(\mathcal{M}(\mathbf{w}^{(t)}) - \mathbf{w}^{(t)}), \tag{1}$$

where $\mathbf{J} := \mathcal{M}'(\mathbf{w}^*)$ is the Jacobian of the mapping $\mathcal{M}$ at $\mathbf{w}^*$. When $\lambda_i := \mathrm{eig}(\mathbf{J}) \in (-1, 1)$, the mapping $\mathcal{M}$ is guaranteed to converge. That is, when $t \to \infty$, $\mathbf{w}^{(t)} \to \mathbf{w}^*$.

It is usually difficult to compute $\mathbf{J}$ for even a simple machine learning model. To alleviate this issue, we can approximate $\mathbf{J}$ with the estimates of its $i$-th eigenvalue $\lambda_i$ by

$$\gamma_i^{(t)} := \frac{\mathcal{M}(\mathbf{w}^{(t)})_i - \mathbf{w}_i^{(t)}}{\mathbf{w}_i^{(t)} - \mathbf{w}_i^{(t-1)}}, \quad \forall i, \tag{2}$$

and extrapolate at each dimension $i$ by:

$$\mathbf{w}_i^{(t+1)} = \mathbf{w}_i^{(t)} + (1 - \gamma_i^{(t)})^{-1}(\mathcal{M}(\mathbf{w}^{(t)})_i - \mathbf{w}_i^{(t)}) . \tag{3}$$

In practice, Aitken's acceleration alternates a step for preparing $\gamma^{(t)}$ and a step for the extrapolation. That is, when $t$ is an even number, $\mathcal{M}$ is used to obtain $\mathbf{w}^{(t+1)}$. Otherwise, the extrapolation (3) is used. A benefit of the above approximation is that the cost for performing an extrapolation is $O(d)$, linear in terms of the dimension.

## 3 Periodic Step-Size Adaptation

When $\mathcal{M}$ is a gradient descent update rule, that is, $\mathcal{M}(\mathbf{w}) \leftarrow \mathbf{w} - \eta \mathbf{g}(\mathbf{w}; \mathsf{D})$, where $\eta$ is a scalar step size, $\mathsf{D}$ is the entire set of training examples, and $\mathbf{g}(\mathbf{w}; \mathsf{D})$ is the gradient of a loss function to be minimized, Aitken's acceleration is equivalent to Newton's method, because

$$\mathbf{J} = \mathcal{M}'(\mathbf{w}) = \mathbf{I} - \eta \mathbf{H}(\mathbf{w}; \mathsf{D}), \tag{4}$$

$$(\mathbf{I} - \mathbf{J})^{-1} = \frac{1}{\eta} \mathbf{H}(\mathbf{w}; \mathsf{D})^{-1}, \text{ and } \mathcal{M}(\mathbf{w}) - \mathbf{w} = \mathbf{w} - \eta \mathbf{g}(\mathbf{w}; \mathsf{D}) - \mathbf{w} = -\eta \mathbf{g}(\mathbf{w}; \mathsf{D}),$$

where $\mathbf{H}(\mathbf{w}; \mathsf{D}) = g'(\mathbf{w}; \mathsf{D})$, the Hessian matrix of the loss function, and the extrapolation given in (1) becomes

$$\mathbf{w} = \mathbf{w} + (\mathbf{I} - \mathbf{J})^{-1}(\mathcal{M}(\mathbf{w}) - \mathbf{w}) = \mathbf{w} - \frac{1}{\eta} \mathbf{H}^{-1} \eta \mathbf{g} = \mathbf{w} - \mathbf{H}^{-1} \mathbf{g}.$$

In this case, Aitken's acceleration enjoys the same local quadratic convergence as Newton's method.

This can also be extended to a SGD update rule: $\mathbf{w}^{(t+1)} \leftarrow \mathbf{w}^{(t)} - \eta \bullet \mathbf{g}(\mathbf{w}^{(t)}; \mathsf{B}^{(t)})$, where the mini-batch $\mathsf{B} \subseteq \mathsf{D}$, $|\mathsf{B}| \ll |\mathsf{D}|$, is a randomly selected small subset of $\mathsf{D}$. A genuine on-line learner usually has $|\mathsf{B}| = 1$. We consider a positive vector-valued step-size $\eta \in \mathbb{R}_+^d$ and "$\bullet$" denotes component-wise (Hadamard) product of two vectors. Again, by exploiting (4), since

$$\mathrm{eig}(\mathbf{I} - \mathrm{diag}(\eta)\mathbf{H}) = \mathrm{eig}(\mathcal{M}') = \mathrm{eig}(\mathbf{J}) \approx \gamma,$$

where $\gamma$ is an estimated eigenvalue of $\mathbf{J}$ as given in (2), when $\mathbf{H}$ is a symmetric matrix, its eigenvalue is given by

$$\mathrm{eig}(\mathbf{J}) = 1 - \eta_i \mathrm{eig}(\mathbf{H}) \Rightarrow \mathrm{eig}(\mathbf{H}) = \frac{1 - \mathrm{eig}(\mathbf{J})}{\eta_i}.$$

Therefore, we can update the step size component-wise by

$$\text{eig}(\mathbf{H}^{-1}) = \frac{\eta_i}{1 - \text{eig}(\mathbf{J})} \approx \frac{\eta_i}{1 - \gamma_i} \Rightarrow \eta_i^{(t+1)} \propto \frac{\eta_i^{(t)}}{1 - \gamma_i^{(t)}}. \tag{5}$$

Since the mapping $\mathcal{M}$ in SGD involves the gradient $\mathbf{g}(\mathbf{w}^{(t)}; \mathsf{B}^{(t)})$ of a randomly selected training example $\mathsf{B}^{(t)}$, $\mathcal{M}$ is itself a random variable. It is unlikely that we can obtain a reliable eigenvalue estimation at each single iteration. To increase stationary of the mapping, we take advantage of the law of large numbers and aggregate consecutive SGD mappings into a new mapping

$$\mathcal{M}^b = \underbrace{\mathcal{M}(\mathcal{M}(\dots \mathcal{M}(\mathbf{w})\dots))}_{b},$$

which reduces the variance of gradient estimation by $\frac{1}{b}$, compared to the plain SGD mapping $\mathcal{M}$. The approximation is valid because $\mathbf{w}^{(t+i)}, i = 0, \dots, b - 1$ are approximately fixed when $\eta$ is sufficiently small [7].

We can proceed to estimate the eigenvalues of $\mathcal{M}^b$ from $\mathbf{w}^{(t)}$, $\mathbf{w}^{(t+b)}$ and $\mathbf{w}^{(t+2b)}$ by applying (2) for each component $i$:

$$\bar{\gamma}_i^b = \frac{\mathbf{w}_i^{(t+2b)} - \mathbf{w}_i^{(t+b)}}{\mathbf{w}_i^{(t+b)} - \mathbf{w}_i^{(t)}} . \tag{6}$$

We note that our aggregate mapping $\mathcal{M}^b$ is different from a mapping that takes $b$ mini-batches as the input in a single iteration. Their difference is similar to that between batch and stochastic gradient descent. Aggregate mappings have $b$ chances to adjust its search direction, while mappings that use $b$ mini-batches together only have one.

With the estimated eigenvalues, we can present the complete update rule to adjust the step size vector $\eta$. To ensure that the estimated values of $\text{eig}(\mathbf{J}) \in (-1, 1)$ and to ensure numerical stability, we introduce a positive constant $\kappa < 1$ as the upper bound of $|\bar{\gamma}_i^b|$. Let $\mathbf{u}$ denote the constrained $\bar{\gamma}^b$. Its components are given by

$$u_i := \text{sgn}(\bar{\gamma}_i^b) \min(|\bar{\gamma}_i^b|, \kappa), \quad \forall i. \tag{7}$$

Then we can update the step size every $2b$ iterations based on $\mathbf{u}$ by:

$$\eta^{(t+2b+1)} = \mathbf{v} \bullet \eta^{(t+2b)}, \tag{8}$$

where $\mathbf{v}$ is a discount factor with components defined by

$$v_i := \frac{m + u_i}{m + \kappa + n}, \quad \forall i. \tag{9}$$

The discount factor is derived from (5) and the fact that when $u < 1$, $\frac{1}{1-u} > e^u \approx 1 + u$ to ensure numerical stability, with $m$ and $n$ controlling the range. Let $\alpha$ be the maximum value and $\beta$ be the minimum value of $v_i$. We can obtain $m$ and $n$ by solving $\beta \leq v_i \leq \alpha$ for all $i$. Since $-\kappa \leq u_i \leq \kappa$, we have $v_i = \alpha$ when $u_i = \kappa$ and $v_i = \beta$ when $u_i = -\kappa$. Solving these equations yields:

$$m = \frac{\alpha + \beta}{\alpha - \beta}\kappa \quad \text{and} \quad n = \frac{2(1 - \alpha)}{\alpha - \beta}\kappa. \tag{10}$$

For example, if we want to set $\alpha = 0.9999$ and $\beta = 0.99$, then $m$ and $n$ will be $201\kappa$ and $0.0202\kappa$, respectively. Setting $0 < \beta < \alpha \leq 1$ ensures that the step size is decreasing and approaches zero so that SGD can be guaranteed to converge [7].

Algorithm 1 shows the PSA algorithm. In a nutshell, PSA applies SGD with a fixed step size and periodically updates the step size by approximating Jacobian of the aggregated mapping. The complexity per iteration is $O(\frac{d}{b})$ because the cost of eigenvalue estimation given in (6) is $2d$ and it is required for every $2b$ iterations. That is, PSA updates $\eta$ after learning from $2b \cdot \mathsf{B}$ examples.

**Algorithm 1** The PSA Algorithm
---
1: **Given:** $\alpha$, $\beta$, $\kappa < 1$ and $b$
2: Initialize $\theta^{(0)}$ and $\eta^{(0)}$; $t \leftarrow 0$; $m \leftarrow \frac{\alpha+\beta}{\alpha-\beta}\kappa$ and $n \leftarrow \frac{2(1-\alpha)}{\alpha-\beta}\kappa$   ▷ Equation (10)
3: **repeat**
4:    Choose a small batch $\mathsf{B}^{(t)}$ uniformly at random from the set of training examples D
5:    update $\theta^{(t+1)} \leftarrow \theta^{(t)} - \eta \bullet \mathbf{g}(\theta^{(t)}; \mathsf{B}^{(t)})$   ▷ SGD update
6:    **if** $(t+1) \bmod 2b = 0$ **then**   ▷ Update $\eta$
7:       update $\bar{\gamma}_i^b \leftarrow \frac{\theta_i^{(t+2b)}-\theta_i^{(t+b)}}{\theta_i^{(t+b)}-\theta_i^{(t)}}$   ▷ Equation (6)
8:       For all $i$, update $u_i \leftarrow \operatorname{sgn}(\bar{\gamma}_i^b) \min(|\bar{\gamma}_i^b|, \kappa)$   ▷ Equation (7)
9:       For all $i$, update $v_i \leftarrow \frac{m+u_i}{m+\kappa+n}$   ▷ Equation (9)
10:       update $\eta^{(t+1)} \leftarrow \mathbf{v} \bullet \eta^{(t)}$   ▷ Equation (8)
11:    **else**
12:       $\eta^{(t+1)} \leftarrow \eta^{(t)}$
13:    **end if**
14:    $t \leftarrow t+1$
15: **until** Convergence
---

## 4   Analysis of PSA

We analyze the accuracy of $\gamma_i^{(t)}$ as an eigenvalue estimate as follows. Let eigen decomposition $\mathbf{J} = \mathbf{Q}\mathbf{\Lambda}\mathbf{Q}^{-1}$ and $\mathbf{u}_i$ be column vectors of $\mathbf{Q}$ and $\mathbf{v}_i^T$ be row vectors of $\mathbf{Q}^{-1}$. Then we have

$$\mathbf{J}^t = \sum_{j=1}^d \lambda_j^t \mathbf{u}_j \mathbf{v}_j^T,$$

where $\lambda_j$ is the $j$-th eigenvalue of $\mathbf{J}$. By applying Taylor's expansion to $\mathcal{M}$, we have

$$
\begin{aligned}
\mathbf{w}^{(t)} - \mathbf{w}^* &\approx \mathbf{J}^t(\mathbf{w}^{(0)} - \mathbf{w}^*) \\
\mathbf{w}^{(t-1)} - \mathbf{w}^* &\approx \mathbf{J}^{t-1}(\mathbf{w}^{(0)} - \mathbf{w}^*) \\
\Rightarrow \Delta^{(t)} = \mathbf{w}^{(t)} - \mathbf{w}^{(t-1)} &\approx \mathbf{J}^t \mathbf{J}^{-1}(\mathbf{J}-\mathbf{I})(\mathbf{w}^{(0)} - \mathbf{w}^*) \\
\Rightarrow \Delta^{(t+1)} = \mathbf{w}^{(t+1)} - \mathbf{w}^{(t)} &\approx \sum_{j=1}^d \lambda_j \lambda_j^t \mathbf{u}_j \mathbf{v}_j^T \mathbf{J}^{-1}(\mathbf{J}-\mathbf{I})(\mathbf{w}^{(0)} - \mathbf{w}^*)
\end{aligned}
$$

Now let

$$\omega_{ij} := \mathbf{e}_i^T \mathbf{u}_j \mathbf{v}_j^T \mathbf{J}^{-1}(\mathbf{J}-\mathbf{I})(\mathbf{w}^{(0)} - \mathbf{w}^*),$$

where $\mathbf{e}_i$ is the $i$-th column of $\mathbf{I}$. Let $\Delta_i$ be the $i$-th element of $\Delta$ and $\lambda_{j_i}$ be the largest eigenvalue of $\mathbf{J}$ such that $\omega_{ij} \neq 0$. Then

$$\gamma_i \equiv \frac{\Delta_i^{(t+1)}}{\Delta_i^{(t)}} = \frac{\sum_{j=1}^d \lambda_j^{t+1}\omega_{ij}}{\sum_{j=1}^d \lambda_j^t \omega_{ij}} = \frac{\lambda_{j_i} + \sum_{j \neq j_i}(\lambda_j/\lambda_{j_i})^t \lambda_j \omega_{ij}/\omega_{ij_i}}{1 + \sum_{j \neq j_i}(\lambda_j/\lambda_{j_i})^t \omega_{ij}/\omega_{ij_i}}.$$

Therefore, we can conclude that

- $\gamma_i \rightarrow \lambda_{j_i}$ as $t \rightarrow \infty$ because $\forall i$, if $\omega_{ij} \neq 0$ then $\lambda_j/\lambda_{j_i} \leq 1$. $\lambda_{j_i} \equiv R_i$ is the $i$-th componentwise rate of convergence.

- $\gamma_i = \lambda_i$ if $\mathbf{J}$ is a diagonal matrix. In this case, our approximation is exact. This happens when there are high percentages of missing data for a Bayesian network model trained by EM [8] and when features are uncorrelated for training a conditional random field model [9].

- $\gamma_i$ is the average of eigenvalues weighted by $\lambda_j^t \omega_{ij}$. Since $\omega_{ij}$ is usually the largest when $i = j$, we have $\gamma_i \approx \lambda_i$.

When we have the least possible step size $\eta^{(t+1)} = \beta\eta^{(t)}$ for all $t \bmod 2b = 0$ in PSA, the expectation of $\mathbf{w}^{(t)}$ obtained by PSA can be shown to be:

$$
\begin{aligned}
E(\mathbf{w}^{(t)}) &= \mathbf{w}^* + \prod_{k=1}^{t}\left(I - \eta^{(0)}\beta^{\lfloor\frac{k}{b}\rfloor}\mathbf{H}(\mathbf{w}^*;\mathrm{D})\right)(\mathbf{w}^{(0)} - \mathbf{w}^*) \\
&= \mathbf{w}^* + \mathbf{S}^{(t)}(\mathbf{w}^{(0)} - \mathbf{w}^*).
\end{aligned}
$$

The rate of convergence is governed by the largest eigenvalue of $\mathbf{S}^{(t)}$. We now derive a bound of this eigenvalue.

**Theorem 1** *Let $\lambda_h$ be the least eigenvalue of $\mathbf{H}(\mathbf{w}^*;\mathrm{D})$. The asymptotic rate of convergence of PSA is bounded by*

$$
eig(\mathbf{S}^{(t)}) \leq \exp\left\{\frac{-\eta^{(0)}\lambda_h b}{1 - \beta}\right\}.
$$

**Proof** We can show that

$$
\begin{aligned}
eig(\mathbf{S}^{(t)}) &= \prod_{k=1}^{t}\left(1 - \eta^{(0)}\beta^{\lfloor\frac{k}{b}\rfloor}\lambda_h\right) \\
&\leq \exp\left\{-\sum_{k=1}^{t}\eta^{(0)}\lambda_h\beta^{\lfloor\frac{k}{b}\rfloor}\right\} = \exp\left\{-\eta^{(0)}\lambda_h\sum_{k=1}^{t}\beta^{\lfloor\frac{k}{b}\rfloor}\right\}
\end{aligned}
$$

because for any $0 \leq a_j < 1$, $1 - a_j \leq e^{-a_j}$,

$$
0 \leq \prod_{j=1}^{n}(1 - a_j) \leq \prod_{j=1}^{n}e^{-a_j} = e^{-\sum_{j=1}^{n}a_j}.
$$

Now, since

$$
\sum_{k=1}^{t}\beta^{\lfloor\frac{k}{b}\rfloor} \approx \left(\sum_{l=0}^{\lfloor\frac{t}{b}\rfloor}b\beta^l\right) = b\sum_{l=0}^{\lfloor\frac{t}{b}\rfloor}\beta^l \longrightarrow \frac{b}{1 - \beta} \quad \text{when } t \to \infty,
$$

we have

$$
eig(\mathbf{S}^{(t)}) \leq \exp\left\{-\eta^{(0)}\lambda_h\sum_{k=1}^{t}\beta^{\lfloor\frac{k}{b}\rfloor}\right\} \to \exp\left\{\frac{-\eta^{(0)}\lambda_h b}{1 - \beta}\right\} \quad \text{when } t \to \infty.
$$

$\square$

Though this analysis suggests that for rapid convergence to $\theta^*$, we should assign $\beta \approx 1$ with a large $b$ and $\eta^{(0)}$, it is based on a worst-case scenario and thus insufficient as a practical guideline for parameter assignment. In practice, we fix $(\alpha, \beta, \kappa) = (0.9999, 0.99, 0.9)$ and tune $b$ as follows. When the training set size $|\mathrm{D}| \gg 2000$, set $b$ in the order of $0.5|\mathrm{D}|/1000$ is usually sufficient. This setting implies that the step size will be adjusted per $|\mathrm{D}|/1000$ examples. In fact, when $b$ is in the same order, PSA performs similarly. Consider the following three settings: $(b, \alpha, \beta) = (10, 0.9999, 0.99)$, $(100, 0.999, 0.9)$ or $(1, 0.99999, 0.999)$. They all yield nearly identical single-pass F-scores for the BaseNP task (see Section 5). The first setting was used in this paper. To see why this is the case, consider the decreasing factor $v_i$ (see (8) and (9)), which will be confined within the interval $(\alpha, \beta)$. Assume that $v_i$ is selected at ransom uniformly, then the mean of $v_i = 0.995$ when $(\alpha, \beta) = (0.9999, 0.99)$ and $\eta_i$ will be decreased by a factor of $0.995$ on average in each PSA update. When $b = 10$, PSA will update $\eta_i$ per 20 examples. After learning from 200 examples, PSA will decrease $\eta_i$ 10 times by a combined factor of $0.9511$. Similarly, we can obtain that the factors for the other two settings are $0.95$ and $0.9512$, respectively, nearly identical.

# 5 Experimental Results

Table 1 shows the tasks chosen for our comparison. The tasks for CRF have been used in competitions and the performance was measured by F-score. **Weight** for CRF reported here is `Number of features` provided by CRF++. **Target** provides the empirical optimal performance achieved by batch learners. If PSA accurately approximates 2SGD, then its single-pass performance should be very close to **Target**. The target F-score for BioNLP/NLPBA is not >85% as reported in [1] because it was due to a bug that included true labels as a feature [1].

Table 1: Tasks for the experiments.

| Task | Model | Training | Test | Tag/Class | Weight | Target |
|------|-------|----------|------|-----------|--------|--------|
| Base NP | CRF | 8936 | 2012 | 3 | 1015662 | 94.0% [10] |
| Chunking | CRF | 8936 | 2012 | 23 | 7448606 | 93.6% [11] |
| BioNLP/NLPBA | CRF | 18546 | 3856 | 11 | 5977675 | 70.0% [12] |
| BioCreative 2 | CRF | 15000 | 5000 | 3 | 10242972 | 86.5% [13] |
| LS FD | LSVM | 2734900 | 2734900 | 2 | 900 | 3.26% |
| LS OCR | LSVM | 1750000 | 1750000 | 2 | 1156 | 23.94% |
| MNIST [14] | CNN | 60000 | 10000 | 10 | 134066 | 0.99% |

## 5.1 Conditional Random Field

We compared PSA with plain SGD and SMD [1] to evaluate PSA's performance for training conditional random fields (CRF). We implemented PSA by replacing the L-BFGS optimizer in CRF++ [11]. For SMD, we used the implementation available in the public domain [2]. Our SGD implementation for CRF is from Bottou [3]. All the above implementations are revisions of CRF++. Finally, we ran the original CRF++ with default settings to obtain the performance results of L-BFGS. We simply used the original parameter settings for SGD and SMD as given in the literature. For PSA, we used $\kappa = 0.9$, $(\alpha, \beta) = (0.9999, 0.99)$, $b = 10$, and $\eta_i^{(0)} = 0.1$, $\forall i$. The batch size is one for all tasks. These parameters were determined by using a small subset from CoNLL 2000 baseNP and we simply used them for all tasks. All of the experiments reported here for CRF were ran on an Intel Q6600 Fedora 8 i686 PC with 4G RAM.

Table 2 compares SGD variants in terms of the execution time and F-scores achieved after processing the training examples for a single pass. Since the loss function in CRF training is convex, the convergence results of L-BFGS can be considered as the empirical minimum. The results show that single-pass F-scores achieved by PSA are about as good as the empirical minima, suggesting that PSA has effectively approximated Hessian in CRF training.

Fig. 1 shows the learning curves in terms of the CPU time. Though as expected, plain SGD is the fastest, it is remarkable that PSA is faster than SMD for all tasks. SMD is supposed to have an edge here because the mini-batch size for SMD was set to 6 or 8, as specified in [1], while PSA used one for all tasks. But PSA is still faster than SMD partly because PSA can take advantage of *the sparsity trick* as plain SGD [15].

## 5.2 Linear SVM

We also evaluated PSA's single-pass performance for training linear SVM. It is straightforward to apply PSA as a primal optimizer for linear SVM. We used two very large data sets: FD (face detection) and OCR (see Table 1), from the Pascal large-scale learning challenge in 2008 and compared the performance of PSA with the state-of-the-art linear SVM solvers: Liblinear 1.33 [16], the winner of the challenge, and SvmSgd, from Bottou's SGD web site. They have been shown to outperform many well-known linear SVM solvers, such as SVM-perf [17] and Pegasos [15].

| | Base NP | | Chunking | | BioNLP/NLPBA | | BioCreative 2 | |
|---|---|---|---|---|---|---|---|---|
| Method (pass) | time | F-score | time | F-score | time | F-score | time | F-score |
| SGD (1) | 1.15 | 92.42 | 13.04 | 92.26 | 12.23 | 66.37 | 3.18 | 34.33 |
| SMD (1) | 41.50 | 91.81 | 350.00 | 91.89 | 522.00 | 66.53 | 497.71 | 69.04 |
| PSA (1) | 16.30 | 93.31 | 160.00 | 93.16 | 206.00 | 69.41 | 191.61 | 80.79 |
| L-BFGS (batch) | 221.17 | 93.91 | 8694.40 | 93.78 | 20130.00 | 70.30 | 1601.50 | 86.82 |

Table 2: CPU time in seconds and F-scores achieved after a single pass of CRF training.

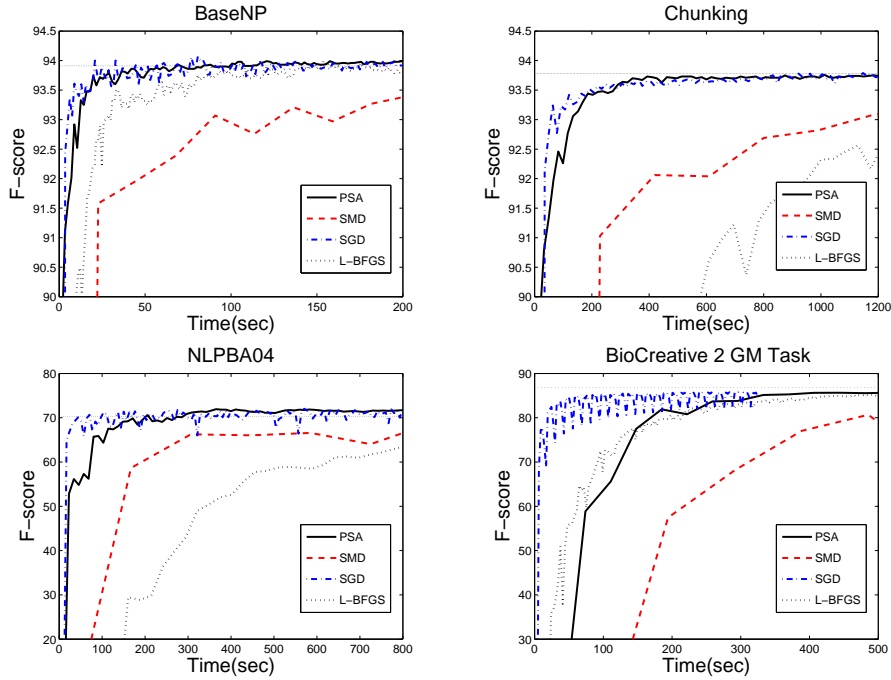

Figure 1: Comparison of CPU time; Horizontal lines indicate target F-scores.

We selected L2-regularized logistic regression as the loss function for PSA and Liblinear because it is twice differentiable. The weight $C$ of the margin error term was set to one. We kept SvmSgd intact. The experiment was run on an Open-SUSE Linux machine with Intel Xeon E7320 CPU (2.13GHz) and 64GB RAM. Table 3 shows the results. Again, PSA achieves the best single-pass accuracy for both tasks. Its test accuracies are very close to that of converged Liblinear. PSA takes much less time than the other two solvers. PSA (1) is faster than SvmSgd (1) for SVM because SvmSgd uses the sparsity trick [15], which speeds up training for sparse data, but otherwise may slow down. Both data sets we used turn out to be dense, *i.e.*, with no zero features. We implemented PSA with the sparsity trick for CRF only but not for SVM and CNN.

| | LS FD | | LS OCR | |
|---|---|---|---|---|
| Method (pass) | accuracy | time | accuracy | time |
| Liblinear converge | 96.74 | 4648.49 | 76.06 | 4454.42 |
| Liblinear (1) | 91.43 | 290.58 | 74.33 | 398.00 |
| SvmSgd (20) | 93.78 | 1135.67 | - | - |
| SvmSgd (10) | 93.77 | 567.68 | 73.71 | 473.35 |
| SvmSgd (1) | 93.60 | 56.78 | 73.76 | 46.96 |
| PSA (1) | 95.10 | 30.65 | 75.68 | 25.33 |

Table 3: Test accuracy rates and elapsed CPU time in seconds by various linear SVM solvers.

The parameter settings for PSA are basically the same as those for CRF but with a large period $b = 1250$ for FD and 500 for OCR. For FD, the worst accuracy by PSA is $94.66\%$ with $b$ between 250 to 2000. For OCR, the worst is $75.20\%$ with $b$ between 100 to 1000, suggesting that PSA is not very sensitive to parameter settings.

### 5.3 Convolutional Neural Network

Approximating Hessian is particularly challenging when the loss function is non-convex. We tested PSA in such a setting by applying PSA to train a large convolutional neural network for the original 10-class MNIST task (see Table 1). We tried to duplicate the implementation of LeNet described in [18] in C++. Our implementation, referred to as "LeNet-S", is a simplified variant of LeNet-5. The differences include that the sub-sampling layers in LeNet-S picks only the upper-left value from a $2 \times 2$ area and abandons the other three. LeNet-S used more maps (50 vs. 16) in the third layer and less nodes (120 vs. 100) in the fifth layer, due to the difference in the previous sub-sampling layer. Finally, we did not implement the Gaussian connections in the last layer. We trained LeNet-S by plain SGD and PSA. The initial $\eta$ for SGD was 0.7 and decreased by 3 percent per pass. For PSA, we used $\kappa = 0.9$, $(\alpha, \beta) = (0.99999, 0.999)$, $b = 10$, $\eta_i^{(0)} = 0.5$, $\forall i$, and the mini-batch size is one for all tasks. We also adapted a trick given in [19] which advises that step sizes in the lower layers should be larger than in the higher layer. Following their trick, the initial step sizes for the first and the third layers were 5 and $\sqrt{2.5}$ times as large as those for the other layers, respectively. The experiments were ran on an Intel Q6600 Fedora 8 i686 PC with 4G RAM.

Table 4 shows the results. To obtain the empirical optimal error rate of our LeNet-S model, we ran plain SGD with sufficient passes and obtained 0.99% error rate at convergence, slightly higher than LeNet-5's 0.95% [18]. Single-pass performance of PSA with the layer trick is within one percentage point to the target. Starting from an initial weight closer to the optimum helped improving PSA's performance further. We ran SGD 100 passes with randomly selected 10K training examples then re-started training with PSA using the rest 50K training examples for a single pass. Though PSA did achieve a better error rate, this is infeasible because it took 4492 seconds to run SGD 100 passes. Finally, though not directly comparable, we also report the performance of TONGA given in [20] as a reference. TONGA is a 2SGD method based on natural gradient.

| Method (pass) | time | error | Method (pass) | time | error |
|---|---|---|---|---|---|
| SGD (1) | 266.77 | 2.36 | PSA w/o layer trick (1) | 311.95 | 2.31 |
| SGD (140) | 37336.20 | 0.99 | PSA w/ layer trick (1) | 311.00 | 1.97 |
| TONGA (n/a) | 500.00 | 2.00 | PSA re-start (1) | 253.72 | 1.90 |

Table 4: CPU time in seconds and percentage test error rates for various neural network trainers.

## 6 Conclusions

It has been shown that given a sufficiently large training set, a single pass of 2SGD generalizes as well as the empirical optimum. Our results show that PSA provides a practical solution to accomplish near optimal performance of 2SGD as predicted theoretically for a variety of large scale models and tasks with a reasonably low cost per iteration compared to competing 2SGD methods. The benefit of 2SGD with PSA over plain SGD becomes clearer when the scale of the tasks are increasingly large. For non-convex neural network tasks, since the curvature of the error surface is so complex, it is still very challenging for an eigenvalue approximation method like PSA. A complete version of this paper will appear as [21]. Source codes of PSA are available at http://aiia.iis.sinica.edu.tw.

## Footnotes

[1] Thanks to Shing-Kit Chan of the Chinese University of Hong Kong for pointing that out.

[2] Available under LGPL from the following URL: `http://sml.nicta.com.au/code/crfsmd/`.

[3] `http://leon.bottou.org/projects/sgd`.

## References

[1] S.V.N. Vishwanathan, Nicol N. Schraudolph, Mark W. Schmidt, and Kevin P. Murphy. Accelerated training of conditional random fields with stochastic gradient methods. In *Proceedings of the 23rd International Conference on Machine Learning (ICML'06)*, Pittsburgh, PA, USA, June 2006.

[2] Michael Collins, Amir Globerson, Terry Koo, Xavier Carreras, and Peter L. Bartlett. Exponentiated gradient algorithms for conditional random fields and max-margin markov networks. *Journal of Machine Learning Research*, 9:1775–1822, August 2008.

[3] Noboru Murata and Shun-Ichi Amari. Statistical analysis of learning dynamics. *Signal Processing*, 74(1):3–28, April 1999.

[4] Léon Bottou and Yann LeCun. On-line learning for very large data sets. *Applied Stochastic Models in Business and Industry*, 21(2):137–151, 2005.

[5] Jorge Nocedal and Stephen J. Wright. *Numerical Optimization*. Springer, 1999.

[6] Léon Bottou. The tradeoffs of large-scale learning. Tutorial, the 21st Annual Conference on Neural Information Processing Systems (NIPS 2007), Vancouver, BC, Canada, December 2007. http://leon.bottou.org/talks/largescale.

[7] Albert Benveniste, Michel Metivier, and Pierre Priouret. *Adaptive Algorithms and Stochastic Approximations*. Springer-Verlag, 1990.

[8] Chun-Nan Hsu, Han-Shen Huang, and Bo-Hou Yang. Global and componentwise extrapolation for accelerating data mining from large incomplete data sets with the EM algorithm. In *Proceedings of the Sixth IEEE International Conference on Data Mining (ICDM'06)*, pages 265–274, Hong Kong, China, December 2006.

[9] Han-Shen Huang, Bo-Hou Yang, Yu-Ming Chang, and Chun-Nan Hsu. Global and componentwise extrapolations for accelerating training of Bayesian networks and conditional random fields. *Data Mining and Knowledge Discovery*, 19(1):58–91, 2009.

[10] Fei Sha and Fernando Pereira. Shallow parsing with conditional random fields. In *Proceedings of Human Language Technology, the North American Chapter of the Association for Computational Linguistics (NAACL'03)*, pages 213–220, 2003.

[11] Taku Kudo. CRF++: Yet another CRF toolkit, 2006. Available under LGPL from the following URL: http://crfpp.sourceforge.net/.

[12] Burr Settles. Biomedical named entity recognition using conditional random fields and novel feature sets. In *Proceedings of the Joint Workshop on Natural Language Processing in Biomedicine and its Applications (JNLPBA-2004)*, pages 104–107, 2004.

[13] Cheng-Ju Kuo, Yu-Ming Chang, Han-Shen Huang, Kuan-Ting Lin, Bo-Hou Yang, Yu-Shi Lin, Chun-Nan Hsu, and I-Fang Chung. Rich feature set, unification of bidirectional parsing and dictionary filtering for high f-score gene mention tagging. In *Proceedings of the Second BioCreative Challenge Evaluation Workshop*, pages 105–107, 2007.

[14] Yann LeCun and Corinna Cortes. The MNIST database of handwritten digits, 1998. http://yann.lecun.com/exdb/mnist/.

[15] Shai Shalev-Shwartz, Yoram Singer, and Nathan Srebro. Pegasos: Primal Estimated sub-GrAdient SOlver for SVM. In *ICML'07: Proceedings of the 24th international conference on Machine learning*, pages 807–814, New York, NY, USA, 2007. ACM Press.

[16] Chih-Chung Chang and Chih-Jen Lin. *LIBSVM: a library for support vector machines*, 2001. Software available at http://www.csie.ntu.edu.tw/~cjlin/libsvm.

[17] Thorsten Joachims. Training linear SVMs in linear time. In *Proceedings of the 12th ACM SIGKDD International Conference on Knowledge Discovery and Data Mining (KDD'06)*, pages 217–226, New York, NY, USA, 2006. ACM.

[18] Yann LeCun, Léon Bottou, Yoshua Bengio, and Patrick Haffner. Gradient-based learning applied to document recognition. *Proceedings of the IEEE*, 86(11):2278–2324, 1998.

[19] Yann LeCun, Leon Bottou, Genevieve B. Orr, and Klaus-Robert Muller. Efficient backprop. In G. Orr and Muller K., editors, *Neural Networks: Tricks of the trade*. Springer, 1998.

[20] Nicolas LeRoux, Pierre-Antoine Manzagol, and Yoshua Bengio. Topmoumoute online natural gradient algorithm. In *Advances in Neural Information Processing Systems, 20 (NIPS 2007)*, Cambridge, MA, USA, 2008. MIT Press.

[21] Chun-Nan Hsu, Yu-Ming Chang, Han-Shen Huang, and Yuh-Jye Lee. Periodic step-size adaptation in second-order gradient descent for single-pass on-line structured learning. To appear in *Mchine Learning, Special Issue on Structured Prediction. DOI: 10.1007/s10994-009-5142-6, 2009*.

